# Fast Computation of Graph Kernels

**S.V. N. Vishwanathan**
`svn.vishwanathan@nicta.com.au`

Statistical Machine Learning, National ICT Australia,
Locked Bag 8001, Canberra ACT 2601, Australia

Research School of Information Sciences & Engineering
Australian National University, Canberra ACT 0200, Australia

**Karsten M. Borgwardt**
`borgwardt@dbs.ifi.lmu.de`

Institute for Computer Science, Ludwig-Maximilians-University Munich
Oettingenstr. 67, 80538 Munich, Germany

**Nicol N. Schraudolph**
`nic.schraudolph@nicta.com.au`

Statistical Machine Learning, National ICT Australia
Locked Bag 8001, Canberra ACT 2601, Australia

Research School of Information Sciences & Engineering
Australian National University, Canberra ACT 0200, Australia

## Abstract

Using extensions of linear algebra concepts to Reproducing Kernel Hilbert Spaces (RKHS), we define a unifying framework for random walk kernels on graphs. Reduction to a Sylvester equation allows us to compute many of these kernels in $O(n^3)$ worst-case time. This includes kernels whose previous worst-case time complexity was $O(n^6)$, such as the geometric kernels of Gärtner et al. [1] and the marginal graph kernels of Kashima et al. [2]. Our algebra in RKHS allow us to exploit sparsity in directed and undirected graphs more effectively than previous methods, yielding sub-cubic computational complexity when combined with conjugate gradient solvers or fixed-point iterations. Experiments on graphs from bioinformatics and other application domains show that our algorithms are often more than 1000 times faster than existing approaches.

## 1 Introduction

Machine learning in domains such as bioinformatics, drug discovery, and web data mining involves the study of relationships between objects. Graphs are natural data structures to model such relations, with nodes representing objects and edges the relationships between them. In this context, one often encounters the question: How similar are two graphs?

Simple ways of comparing graphs which are based on pairwise comparison of nodes or edges, are possible in quadratic time, yet may neglect information represented by the *structure* of the graph. Graph kernels, as originally proposed by Gärtner et al. [1], Kashima et al. [2], Borgwardt et al. [3], take the structure of the graph into account. They work by *counting* the number of common random walks between two graphs. Even though the number of common random walks could potentially be

exponential, polynomial time algorithms exist for computing these kernels. Unfortunately for the practitioner, these kernels are still prohibitively expensive since their computation scales as $O(n^6)$, where $n$ is the number of vertices in the input graphs. This severely limits their applicability to large-scale problems, as commonly found in areas such as bioinformatics.

In this paper, we extend common concepts from linear algebra to Reproducing Kernel Hilbert Spaces (RKHS), and use these extensions to define a unifying framework for random walk kernels. We show that computing many random walk graph kernels including those of Gärtner et al. [1] and Kashima et al. [2] can be reduced to the problem of solving a large linear system, which can then be solved efficiently by a variety of methods which exploit the structure of the problem.

## 2 Extending Linear Algebra to RKHS

Let $\phi : \mathcal{X} \to \mathcal{H}$ denote the feature map from an input space $\mathcal{X}$ to the RKHS $\mathcal{H}$ associated with the kernel $\kappa(x, x') = \langle \phi(x), \phi(x') \rangle_{\mathcal{H}}$. Given an $n$ by $m$ matrix $X \in \mathcal{X}^{n \times m}$ of elements $X_{ij} \in \mathcal{X}$, we extend $\phi$ to matrix arguments by defining $\Phi : \mathcal{X}^{n \times m} \to \mathcal{H}^{n \times m}$ via $[\Phi(X)]_{ij} := \phi(X_{ij})$. We can now borrow concepts from tensor calculus to extend certain linear algebra operations to $\mathcal{H}$:

**Definition 1** Let $A \in \mathcal{X}^{n \times m}$, $B \in \mathcal{X}^{m \times p}$, and $C \in \mathbb{R}^{m \times p}$. The matrix products $\Phi(A)\Phi(B) \in \mathbb{R}^{n \times p}$ and $\Phi(A)\, C \in \mathcal{H}^{n \times p}$ are

$$[\Phi(A)\Phi(B)]_{ik} := \sum_j \langle \phi(A_{ij}), \phi(B_{jk}) \rangle_{\mathcal{H}} \quad and \quad [\Phi(A)\, C]_{ik} := \sum_j \phi(A_{ij})\, C_{jk}.$$

Given $A \in \mathbb{R}^{n \times m}$ and $B \in \mathbb{R}^{p \times q}$ the Kronecker product $A \otimes B \in \mathbb{R}^{np \times mq}$ and vec operator are defined as

$$A \otimes B := \begin{bmatrix} A_{11}B & A_{12}B & \dots & A_{1m}B \\ \vdots & \vdots & \vdots & \vdots \\ A_{n1}B & A_{n2}B & \dots & A_{nm}B \end{bmatrix}, \quad \mathrm{vec}(A) := \begin{bmatrix} A_{*1} \\ \vdots \\ A_{*m} \end{bmatrix}, \quad (1)$$

where $A_{*j}$ denotes the $j$-th column of $A$. They are linked by the well-known property:

$$\mathrm{vec}(ABC) = (C^\top \otimes A)\, \mathrm{vec}(B). \quad (2)$$

**Definition 2** Let $A \in \mathcal{X}^{n \times m}$ and $B \in \mathcal{X}^{p \times q}$. The Kronecker product $\Phi(A) \otimes \Phi(B) \in \mathbb{R}^{np \times mq}$ is

$$[\Phi(A) \otimes \Phi(B)]_{ip+k, jq+l} := \langle \phi(A_{ij}), \phi(B_{kl}) \rangle_{\mathcal{H}}. \quad (3)$$

It is easily shown that the above extensions to RKHS obey an analogue of (2):

**Lemma 1** If $A \in \mathcal{X}^{n \times m}$, $B \in \mathbb{R}^{m \times p}$, and $C \in \mathcal{X}^{p \times q}$, then

$$\mathrm{vec}(\Phi(A)\, B\, \Phi(C)) = (\Phi(C)^\top \otimes \Phi(A))\, \mathrm{vec}(B). \quad (4)$$

If $p = q = n = m$, direct computation of the right hand side of (4) requires $O(n^4)$ kernel evaluations. For an arbitrary kernel the left hand side also requires a similar effort. But, if the RKHS $\mathcal{H}$ is isomorphic to $\mathbb{R}^r$, in other words the feature map $\phi(\cdot) \in \mathbb{R}^r$, the left hand side of (4) is easily computed in $O(n^3 r)$ operations. Our efficient computation schemes described in Section 4 will exploit this observation.

## 3 Random Walk Kernels

Random walk kernels on graphs are based on a simple idea: Given a pair of graphs perform a random walk on both of them and *count* the number of *matching* walks [1, 2, 3]. These kernels mainly differ in the way the similarity between random walks is computed. For instance, Gärtner et al. [1] count the number of nodes in the random walk which have the same label. They also include a decay factor to ensure convergence. Kashima et al. [2], and Borgwardt et al. [3] on the other hand, use a kernel defined on nodes and edges in order to compute similarity between random walks, and define an initial probability distribution over nodes in order to ensure convergence. In this section we present a unifying framework which includes the above mentioned kernels as special cases.

## 3.1 Notation

We use $\mathbf{e}_i$ to denote the $i$-th standard basis (*i.e.,* a vector of all zeros with the $i$-th entry set to one), $\mathbf{e}$ to denote a vector with all entries set to one, $\mathbf{0}$ to denote the vector of all zeros, and $\mathbf{I}$ to denote the identity matrix. When it is clear from context we will not mention the dimensions of these vectors and matrices.

A graph $G \in \mathcal{G}$ consists of an ordered and finite set of $n$ vertices $V$ denoted by $\{v_1, v_2, \ldots, v_n\}$, and a finite set of edges $E \subset V \times V$. A vertex $v_i$ is said to be a neighbor of another vertex $v_j$ if they are connected by an edge. $G$ is said to be undirected if $(v_i, v_j) \in E \iff (v_j, v_i) \in E$ for all edges. The unnormalized adjacency matrix of $G$ is an $n \times n$ real matrix $P$ with $P_{ij} = 1$ if $(v_i, v_j) \in E$, and $0$ otherwise. If $G$ is weighted then $P$ can contain non-negative entries other than zeros and ones, *i.e.,* $P_{ij} \in (0, \infty)$ if $(v_i, v_j) \in E$ and zero otherwise.

Let $D$ be an $n \times n$ diagonal matrix with entries $D_{ii} = \sum_j P_{ij}$. The matrix $A := PD^{-1}$ is then called the normalized adjacency matrix, or simply adjacency matrix. A walk $w$ on $G$ is a sequence of indices $w_1, w_2, \ldots w_{t+1}$ where $(v_{w_i}, v_{w_{i+1}}) \in E$ for all $1 \le i \le t$. The length of a walk is equal to the number of edges encountered during the walk (here: $t$). A graph is said to be connected if any two pairs of vertices can be connected by a walk; here we always work with connected graphs. A random walk is a walk where $\mathbb{P}(w_{i+1}|w_1, \ldots w_i) = A_{w_i, w_{i+1}}$, *i.e.,* the probability at $w_i$ of picking $w_{i+1}$ next is directly proportional to the weight of the edge $(v_{w_i}, v_{w_{i+1}})$. The $t$-th power of the transition matrix $A$ describes the probability of $t$-length walks. In other words, $[A^t]_{ij}$ denotes the probability of a transition from vertex $v_i$ to vertex $v_j$ via a walk of length $t$. We use this intuition to define random walk kernels on graphs.

Let $\mathcal{X}$ be a set of labels which includes the special label $\epsilon$. Every edge labeled graph $G$ is associated with a label matrix $L \in \mathcal{X}^{n \times n}$, such that $L_{ij} = \epsilon$ iff $(v_i, v_j) \notin E$, in other words only those edges which are present in the graph get a non-$\epsilon$ label. Let $\mathcal{H}$ be the RKHS endowed with the kernel $\kappa : \mathcal{X} \times \mathcal{X} \to \mathbb{R}$, and let $\phi : \mathcal{X} \to \mathcal{H}$ denote the corresponding feature map which maps $\epsilon$ to the zero element of $\mathcal{H}$. We use $\Phi(L)$ to denote the feature matrix of $G$. For ease of exposition we do not consider labels on vertices here, though our results hold for that case as well. Henceforth we use the term labeled graph to denote an edge-labeled graph.

## 3.2 Product Graphs

Given two graphs $G(V, E)$ and $G'(V', E')$, the product graph $G_\times(V_\times, E_\times)$ is a graph with $nn'$ vertices, each representing a pair of vertices from $G$ and $G'$, respectively. An edge exists in $E_\times$ iff the corresponding vertices are adjacent in both $G$ and $G'$. Thus

$$V_\times = \{(v_i, v'_{i'}) : v_i \in V \land v'_{i'} \in V'\}, \tag{5}$$

$$E_\times = \{((v_i, v'_{i'}), (v_j, v'_{j'})) : (v_i, v_j) \in E \land (v'_{i'}, v'_{j'}) \in E'\}. \tag{6}$$

If $A$ and $A'$ are the adjacency matrices of $G$ and $G'$, respectively, the adjacency matrix of the product graph $G_\times$ is $A_\times = A \otimes A'$. An edge exists in the product graph iff an edge exits in both $G$ and $G'$, therefore performing a simultaneous random walk on $G$ and $G'$ is equivalent to performing a random walk on the product graph [4].

Let $p$ and $p'$ denote initial probability distributions over vertices of $G$ and $G'$. Then the initial probability distribution $p_\times$ of the product graph is $p_\times := p \otimes p'$. Likewise, if $q$ and $q'$ denote stopping probabilities (*i.e.,* the probability that a random walk ends at a given vertex), the stopping probability $q_\times$ of the product graph is $q_\times := q \otimes q'$.

If $G$ and $G'$ are edge-labeled, we can associate a weight matrix $W_\times \in \mathbb{R}^{nn' \times nn'}$ with $G_\times$, using our Kronecker product in RKHS (Definition 2): $W_\times = \Phi(L) \otimes \Phi(L')$. As a consequence of the definition of $\Phi(L)$ and $\Phi(L')$, the entries of $W_\times$ are non-zero only if the corresponding edge exists in the product graph. The weight matrix is closely related to the adjacency matrix: assume that $\mathcal{H} = \mathbb{R}$ endowed with the usual dot product, and $\phi(L_{ij}) = 1$ if $(v_i, v_j) \in E$ or zero otherwise. Then $\Phi(L) = A$ and $\Phi(L') = A'$, and consequently $W_\times = A_\times$, *i.e.,* the weight matrix is identical to the adjacency matrix of the product graph.

To extend the above discussion, assume that $\mathcal{H} = \mathbb{R}^d$ endowed with the usual dot product, and that there are $d$ distinct edge labels $\{1, 2, \ldots, d\}$. For each edge $(v_i, v_j) \in E$ we have $\phi(L_{ij}) = \mathbf{e}_l$ if

the edge $(v_i, v_j)$ is labeled $l$. All other entries of $\Phi(L)$ are set to $\mathbf{0}$. $\kappa$ is therefore a delta kernel, *i.e.,* its value between any two edges is one iff the labels on the edges match, and zero otherwise. The weight matrix $W_\times$ has a non-zero entry iff an edge exists in the product graph and the corresponding edges in $G$ and $G'$ have the same label. Let $^lA$ denote the adjacency matrix of the graph filtered by the label $l$, *i.e.,* $^lA_{ij} = A_{ij}$ if $L_{ij} = l$ and zero otherwise. Some simple algebra (omitted for the sake of brevity) shows that the weight matrix of the product graph can be written as

$$ W_\times = \sum_{l=1}^{d} {}^lA \otimes {}^lA'. \tag{7} $$

## 3.3 Kernel Definition

Performing a random walk on the product graph $G_\times$ is equivalent to performing a simultaneous random walk on the graphs $G$ and $G'$ [4]. Therefore, the $(in+j, i'n'+j')$-th entry of $A_\times^k$ represents the probability of simultaneous $k$ length random walks on $G$ (starting from vertex $v_i$ and ending in vertex $v_j$) and $G'$ (starting from vertex $v'_{i'}$ and ending in vertex $v'_{j'}$). The entries of $W_\times$ represent similarity between edges. The $(in+j, i'n'+j')$-th entry of $W_\times^k$ represents the similarity between simultaneous $k$ length random walks on $G$ and $G'$ measured via the kernel function $\kappa$.

Given the weight matrix $W_\times$, initial and stopping probability distributions $p_\times$ and $q_\times$, and an appropriately chosen discrete measure $\mu$, we can define a random walk kernel on $G$ and $G'$ as

$$ k(G, G') := \sum_{k=0}^{\infty} \mu(k)\, q_\times^\top W_\times^k p_\times. \tag{8} $$

In order to show that (8) is a valid Mercer kernel we need the following technical lemma.

**Lemma 2** $\forall\, k \in \mathbb{N}_0 : \ W_\times^k p_\times = \text{vec}[\Phi(L')^k p'\, (\Phi(L)^k p)^\top]$.

**Proof** By induction over $k$. Base case: $k = 0$. Since $\Phi(L')^0 = \Phi(L)^0 = \mathbf{I}$, using (2) we can write

$$ W_\times^0 p_\times = p_\times = (p \otimes p')\,\text{vec}(1) = \text{vec}(p'\, 1\, p^\top) = \text{vec}[\Phi(L')^0 p'\, (\Phi(L)^0 p)^\top]. $$

Induction from $k$ to $k+1$: Using Lemma 1 we obtain

$$ W_\times^{k+1} p_\times = W_\times W_\times^k p_\times = (\Phi(L) \otimes \Phi(L'))\,\text{vec}[\Phi(L')^k p'\, (\Phi(L)^k p)^\top] $$
$$ = \text{vec}[\Phi(L')\Phi(L')^k p'\, (\Phi(L)^k p)^\top \Phi(L)^\top] = \text{vec}[\Phi(L')^{k+1} p'\, (\Phi(L)^{k+1} p)^\top]. \qquad \blacksquare $$

**Lemma 3** *If the measure* $\mu(k)$ *is such that* (8) *converges, then it defines a valid Mercer kernel.*

**Proof** Using Lemmas 1 and 2 we can write

$$ q_\times^\top W_\times^k p_\times = (q \otimes q')\,\text{vec}[\Phi(L')^k p'\, (\Phi(L)^k p)^\top] = \text{vec}[q'^\top \Phi(L')^k p'\, (\Phi(L)^k p)^\top q] $$
$$ = \underbrace{(q^\top \Phi(L)^k p)^\top}_{\psi_k(G)^\top}\, \underbrace{(q'^\top \Phi(L')^k p')}_{\psi_k(G')}. $$

Each individual term of (8) equals $\psi_k(G)^\top \psi_k(G')$ for some function $\psi_k$, and is therefore a valid kernel. The lemma follows since a convex combination of kernels is itself a valid kernel. $\qquad \blacksquare$

## 3.4 Special Cases

A popular choice to ensure convergence of (8) is to assume $\mu(k) = \lambda^k$ for some $\lambda > 0$. If $\lambda$ is sufficiently small[1] then (8) is well defined, and we can write

$$ k(G, G') = \sum_k \lambda^k q_\times^\top W_\times^k p_\times = q_\times^\top (\mathbf{I} - \lambda W_\times)^{-1} p_\times. \tag{9} $$

Kashima et al. [2] use marginalization and probabilities of random walks to define kernels on graphs. Given transition probability matrices $P$ and $P'$ associated with graphs $G$ and $G'$ respectively, their kernel can be written as (see Eq. 1.19, [2])

$$ k(G, G') = q_\times^\top (\mathbf{I} - T_\times)^{-1} p_\times, \tag{10} $$

where $T_\times := (\text{vec}(P)\,\text{vec}(P')^\top) \odot (\Phi(L) \otimes \Phi(L'))$, using $\odot$ to denote element-wise (Hadamard) multiplication. The edge kernel $\hat\kappa(L_{ij}, L'_{i'j'}) := P_{ij} P'_{i'j'} \kappa(L_{ij}, L'_{i,j'})$ with $\lambda = 1$ recovers (9).

Gärtner et al. [1] use the adjacency matrix of the product graph to define the so-called geometric kernel

$$k(G, G') = \sum_{i=1}^{n} \sum_{j=1}^{n'} \sum_{k=0}^{\infty} \lambda^k [A_\times^k]_{ij}. \tag{11}$$

To recover their kernel in our framework, assume an uniform distribution over the vertices of $G$ and $G'$, *i.e.*, set $p = q = 1/n$ and $p' = q' = 1/n'$. The initial as well as final probability distribution over vertices of $G_\times$ is given by $p_\times = q_\times = \mathbf{e}\,/(nn')$. Setting $\Phi(L) := A$, and hence $\Phi(L') = A'$ and $W_\times = A_\times$, we can rewrite (8) to obtain

$$k(G, G') = \sum_{k=0}^{\infty} \lambda^k q_\times^\top A_\times^k p_\times = \frac{1}{n^2 n'^2} \sum_{i=1}^{n} \sum_{j=1}^{n'} \sum_{k=0}^{\infty} \lambda^k [A_\times^k]_{ij},$$

which recovers (11) to within a constant factor.

# 4 Efficient Computation

In this section we show that iterative methods, including those based on Sylvester equations, conjugate gradients, and fixed-point iterations, can be used to greatly speed up the computation of (9).

## 4.1 Sylvester Equation Methods

Consider the following equation, commonly known as the Sylvester or Lyapunov equation:

$$X = SXT + X_0. \tag{12}$$

Here, $S, T, X_0 \in \mathbb{R}^{n \times n}$ are given and we need for solve for $X \in \mathbb{R}^{n \times n}$. These equations can be readily solved in $O(n^3)$ time with freely available code [5], *e.g.* Matlab's `dlyap` method. The generalized Sylvester equation

$$X = \sum_{i=1}^{d} S_i X T_i + X_0 \tag{13}$$

can also be solved efficiently, albeit at a slightly higher computational cost of $O(dn^3)$.

We now show that if the weight matrix $W_\times$ can be written as (7) then the problem of computing the graph kernel (9) can be reduced to the problem of solving the following Sylvester equation:

$$X = \sum_i {}^{i}A' \lambda X \,{}^{i}A^\top + X_0, \tag{14}$$

where $\text{vec}(X_0) = p_\times$. We begin by *flattening* the above equation:

$$\text{vec}(X) = \lambda \sum_i \text{vec}({}^{i}A' X \,{}^{i}A^\top) + p_\times. \tag{15}$$

Using Lemma 1 we can rewrite (15) as
$$\left(\mathbf{I} - \lambda \sum_i {}^{i}A \otimes {}^{i}A'\right) \text{vec}(X) = p_\times, \tag{16}$$

use (7), and solve for $\text{vec}(X)$:
$$\text{vec}(X) = (\mathbf{I} - \lambda W_\times)^{-1} p_\times. \tag{17}$$

Multiplying both sides by $q_\times^\top$ yields
$$q_\times^\top \text{vec}(X) = q_\times^\top (\mathbf{I} - \lambda W_\times)^{-1} p_\times. \tag{18}$$

The right-hand side of (18) is the graph kernel (9). Given the solution $X$ of the Sylvester equation (14), the graph kernel can be obtained as $q_\times^\top \text{vec}(X)$ in $O(n^2)$ time. Since solving the generalized Sylvester equation takes $O(dn^3)$ time, computing the graph kernel in this fashion is significantly faster than the $O(n^6)$ time required by the direct approach.

Where the number of labels $d$ is large, the computational cost may be reduced further by computing matrices $S$ and $T$ such that $W_\times \approx S \otimes T$. We then simply solve the simple Sylvester equation (12) involving these matrices. Finding the nearest Kronecker product approximating a matrix such as $W_\times$ is a well-studied problem in numerical linear algebra and efficient algorithms which exploit sparsity of $W_\times$ are readily available [6].

## 4.2 Conjugate Gradient Methods

Given a matrix $M$ and a vector $b$, conjugate gradient (CG) methods solve the system of equations $Mx = b$ efficiently [7]. While they are designed for symmetric positive semi-definite matrices, CG solvers can also be used to solve other linear systems efficiently. They are particularly efficient if the matrix is rank deficient, or has a small *effective rank*, *i.e.,* number of distinct eigenvalues. Furthermore, if computing matrix-vector products is cheap — because $M$ is sparse, for instance — the CG solver can be sped up significantly [7]. Specifically, if computing $Mv$ for an arbitrary vector $v$ requires $O(k)$ time, and the effective rank of the matrix is $m$, then a CG solver requires only $O(mk)$ time to solve $Mx = b$.

The graph kernel (9) can be computed by a two-step procedure: First we solve the linear system

$$(\mathbf{I} - \lambda W_\times)\, x = p_\times, \tag{19}$$

for $x$, then we compute $q_\times^\top x$. We now focus on efficient ways to solve (19) with a CG solver. Recall that if $G$ and $G'$ contain $n$ vertices each then $W_\times$ is a $n^2 \times n^2$ matrix. Directly computing the matrix-vector product $W_\times r$, requires $O(n^4)$ time. Key to our speed-ups is the ability to exploit Lemma 1 to compute this matrix-vector product more efficiently: Recall that $W_\times = \Phi(L) \otimes \Phi(L')$. Letting $r = \text{vec}(R)$, we can use Lemma 1 to write

$$W_\times r = (\Phi(L) \otimes \Phi(L'))\, \text{vec}(R) = \text{vec}(\Phi(L')R\,\Phi(L)^\top). \tag{20}$$

If $\phi(\cdot) \in \mathbb{R}^r$ for some $r$, then the above matrix-vector product can be computed in $O(n^3 r)$ time. If $\Phi(L)$ and $\Phi(L')$ are sparse, however, then $\Phi(L')R\,\Phi(L)^\top$ can be computed yet more efficiently: if there are $O(n)$ non-$\epsilon$ entries in $\Phi(L)$ and $\Phi(L')$, then computing (20) requires only $O(n^2)$ time.

## 4.3 Fixed-Point Iterations

Fixed-point methods begin by rewriting (19) as $\qquad x = p_\times + \lambda W_\times x. \tag{21}$

Now, solving for $x$ is equivalent to finding a fixed point of the above iteration [7]. Letting $x_t$ denote the value of $x$ at iteration $t$, we set $x_0 := p_\times$, then compute

$$x_{t+1} = p_\times + \lambda W_\times x_t \tag{22}$$

repeatedly until $||x_{t+1} - x_t|| < \varepsilon$, where $|| \cdot ||$ denotes the Euclidean norm and $\varepsilon$ some pre-defined tolerance. This is guaranteed to converge if all eigenvalues of $\lambda W_\times$ lie inside the unit disk; this can be ensured by setting $\lambda < 1/\xi_{\max}$, where $\xi_{\max}$ is the largest-magnitude eigenvalue of $W_\times$.

The above is closely related to the power method used to compute the largest eigenvalue of a matrix [8]; efficient preconditioners can also be used to speed up convergence [8]. Since each iteration of (22) involves computation of the matrix-vector product $W_\times x_t$, all speed-ups for computing the matrix-vector product discussed in Section 4.2 are applicable here. In particular, we exploit the fact that $W_\times$ is a sum of Kronecker products to reduce the worst-case time complexity to $O(n^3)$ in our experiments, in contrast to Kashima et al. [2] who computed the matrix-vector product explicitly.

# 5 Experiments

To assess the practical impact of our algorithmic improvements, we compared our techniques from Section 4 with Gärtner et al.'s [1] direct approach as a baseline. All code was written in MATLAB Release 14, and experiments run on a 2.6 GHz Intel Pentium 4 PC with 2 GB of main memory running Suse Linux. The Matlab function `dlyap` was used to solve the Sylvester equation.

By default, we used a value of $\lambda = 0.001$, and set the tolerance for both CG solver and fixed-point iteration to $10^{-6}$ for all our experiments. We used Lemma 1 to speed up matrix-vector multiplication for both CG and fixed-point methods (*cf.* Section 4.2). Since all our methods are exact and produce the same kernel values (to numerical precision), we only report their runtimes below.

We tested the practical feasibility of the presented techniques on four real-world datasets whose size mandates fast graph kernel computation; two datasets of molecular compounds (MUTAG and PTC), and two datasets with hundreds of graphs describing protein tertiary structure (Protein and Enzyme). Graph kernels provide useful measures of similarity for all these graphs; please refer to the addendum for more details on these datasets, and applications for graph kernels on them.

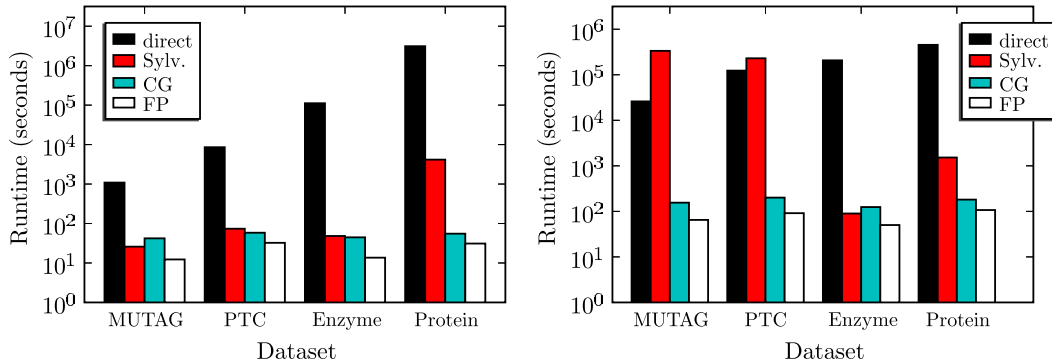

Figure 1: Time (in seconds on a log-scale) to compute $100 \times 100$ kernel matrix for unlabeled (left) *resp.* labelled (right) graphs from several datasets. Compare the conventional direct method (black) to our fast Sylvester equation, conjugate gradient (CG), and fixed-point iteration (FP) approaches.

## 5.1 Unlabeled Graphs

In a first series of experiments, we compared graph topology only on our 4 datasets, *i.e.,* without considering node and edge labels. We report the time taken to compute the full graph kernel matrix for various sizes (number of graphs) in Table 1 and show the results for computing a $100 \times 100$ sub-matrix in Figure 1 (left).

On unlabeled graphs, conjugate gradient and fixed-point iteration — sped up via our Lemma 1 — are consistently about two orders of magnitude faster than the conventional direct method. The Sylvester approach is very competitive on smaller graphs (outperforming CG on MUTAG) but slows down with increasing number of nodes per graph; this is because we were unable to incorporate Lemma 1 into Matlab's black-box `dlyap` solver. Even so, the Sylvester approach still greatly outperforms the direct method.

## 5.2 Labeled Graphs

In a second series of experiments, we compared graphs with node and edge labels. On our two protein datasets we employed a linear kernel to measure similarity between edge labels representing distances (in ångströms) between secondary structure elements. On our two chemical datasets we used a delta kernel to compare edge labels reflecting types of bonds in molecules. We report results in Table 2 and Figure 1 (right).

On labeled graphs, our three methods outperform the direct approach by about a factor of 1000 when using the linear kernel. In the experiments with the delta kernel, conjugate gradient and fixed-point iteration are still at least two orders of magnitude faster. Since we did not have access to a generalized Sylvester equation (13) solver, we had to use a Kronecker product approximation [6] which dramatically slowed down the Sylvester equation approach.

Table 1: Time to compute kernel matrix for given number of unlabeled graphs from various datasets.

| dataset | MUTAG | | PTC | | Enzyme | | Protein | |
|---|---|---|---|---|---|---|---|---|
| nodes/graph | 17.7 | | 26.7 | | 32.6 | | 38.6 | |
| edges/node | 2.2 | | 1.9 | | 3.8 | | 3.7 | |
| #graphs | 100 | 230 | 100 | 417 | 100 | 600 | 100 | 1128 |
| Direct | 18'09" | 104'31" | 142'53" | 41h* | 31h* | 46.5d* | 36d* | 12.5y* |
| Sylvester | 25.9" | 2'16" | 73.8" | 19'30" | 48.3" | 36'43" | 69'15" | 6.1d* |
| Conjugate | 42.1" | 4'04" | 58.4" | 19'27" | 44.6" | 34'58" | 55.3" | 97'13" |
| Fixed-Point | 12.3" | 1'09" | 32.4" | 5'59" | 13.6" | 15'23" | 31.1" | 40'58" |

∗: Extrapolated; run did not finish in time available.

Table 2: Time to compute kernel matrix for given number of labeled graphs from various datasets.

| kernel | delta | | | | linear | | | |
|---|---|---|---|---|---|---|---|---|
| dataset | MUTAG | | PTC | | Enzyme | | Protein | |
| #graphs | 100 | 230 | 100 | 417 | 100 | 600 | 100 | 1128 |
| Direct | 7.2h | 1.6d* | 1.4d* | 25d* | 2.4d* | 86d* | 5.3d* | 18y* |
| Sylvester | 3.9d* | 21d* | 2.7d* | 46d* | 89.8" | 53'55" | 25'24" | 2.3d* |
| Conjugate | 2'35" | 13'46" | 3'20" | 53'31" | 124.4" | 71'28" | 3'01" | 4.1h |
| Fixed Point | 1'05" | 6'09" | 1'31" | 26'52" | 50.1" | 35'24" | 1'47" | 1.9h |

∗: Extrapolated; run did not finish in time available.

## 6 Outlook and Discussion

We have shown that computing random walk graph kernels is essentially equivalent to solving a large linear system. We have extended a well-known identity for Kronecker products which allows us to exploit the structure inherent in this problem. From this we have derived three efficient techniques to solve the linear system, employing either Sylvester equations, conjugate gradients, or fixed-point iterations. Experiments on real-world datasets have shown our methods to be scalable and fast, in some instances outperforming the conventional approach by more than three orders of magnitude.

Even though the Sylvester equation method has a worst-case complexity of $O(n^3)$, the conjugate gradient and fixed-point methods tend to be faster on all our datasets. This is because computing matrix-vector products via Lemma 1 is quite efficient when the graphs are sparse, so that the feature matrices $\Phi(L)$ and $\Phi(L')$ contain only $O(n)$ non-$\epsilon$ entries. Matlab's black-box `dlyap` solver is unable to exploit this sparsity; we are working on more capable alternatives. An efficient generalized Sylvester solver requires extensive use of tensor calculus and is part of ongoing work.

As more and more graph-structured data becomes available in areas such as biology, web data mining, *etc.*, graph classification will gain importance over coming years. Hence there is a pressing need to speed up the computation of similarity metrics on graphs. We have shown that sparsity, low effective rank, and Kronecker product structure can be exploited to greatly reduce the computational cost of graph kernels; taking advantage of other forms of structure in $W_\times$ remains a challenge. Now that the computation of random walk graph kernels is viable for practical problem sizes, it will open the doors for their application in hitherto unexplored domains. The algorithmic challenge now is how to integrate higher-order structures, such as spanning trees, in graph comparisons.

### Acknowledgments

National ICT Australia is funded by the Australian Government's Department of Communications, Information Technology and the Arts and the Australian Research Council through Backing Australia's Ability and the ICT Center of Excellence program. This work is supported by the IST Program of the European Community, under the Pascal Network of Excellence, IST-2002-506778, and by the German Ministry for Education, Science, Research and Technology (BMBF) under grant no. 031U112F within the BFAM (Bioinformatics for the Functional Analysis of Mammalian Genomes) project, part of the German Genome Analysis Network (NGFN).

## Footnotes

[1]The values of $\lambda$ which ensure convergence depends on the spectrum of $W_\times$.

## References

[1] T. Gärtner, P. Flach, and S. Wrobel. On graph kernels: Hardness results and efficient alternatives. In B. Schölkopf and M. K. Warmuth, editors, *Proc. Annual Conf. Comput. Learning Theory*. Springer, 2003.

[2] H. Kashima, K. Tsuda, and A. Inokuchi. Kernels on graphs. In K. Tsuda, B. Schölkopf, and J. Vert, editors, *Kernels and Bioinformatics*, Cambridge, MA, 2004. MIT Press.

[3] K. M. Borgwardt, C. S. Ong, S. Schonauer, S. V. N. Vishwanathan, A. J. Smola, and H. P. Kriegel. Protein function prediction via graph kernels. *Bioinformatics*, 21(Suppl 1):i47–i56, 2005.

[4] F. Harary. *Graph Theory*. Addison-Wesley, Reading, MA, 1969.

[5] J. D. Gardiner, A. L. Laub, J. J. Amato, and C. B. Moler. Solution of the Sylvester matrix equation $AXB^\top + CXD^\top = E$. *ACM Transactions on Mathematical Software*, 18(2):223–231, 1992.

[6] C. F. V. Loan. The ubiquitous kronecker product. *Journal of Computational and Applied Mathematics*, 123:85 – 100, 2000.

[7] J. Nocedal and S. J. Wright. *Numerical Optimization*. Springer Series in Operations Research, 1999.

[8] G. H. Golub and C. F. Van Loan. *Matrix Computations*. John Hopkins University Press, Baltimore, MD, 3rd edition, 1996.
